# A Model of Feedback to the Lateral Geniculate Nucleus

**Carlos D. Brody**
Computation and Neural Systems Program
California Institute of Technology
Pasadena, CA 91125

## Abstract

Simplified models of the lateral geniculate nucles (LGN) and striate cortex illustrate the possibility that feedback to the LGN may be used for robust, low-level pattern analysis. The information fed back to the LGN is rebroadcast to cortex using the LGN's full fan-out, so the cortex→LGN→cortex pathway mediates extensive cortico-cortical communication while keeping the number of necessary connections small.

## 1  INTRODUCTION

The lateral geniculate nucleus (LGN) in the thalamus is often considered as just a relay station on the way from the retina to visual cortex, since receptive field properties of neurons in the LGN are very similar to retinal ganglion cell receptive field properties. However, there is a massive projection from cortex back to the LGN: it is estimated that 3-4 times more synapses in the LGN are due to corticogeniculate connections than those due to retinogeniculate connections [12]. This suggests some important processing role for the LGN, but the nature of the computation performed has remained far from clear.

I will first briefly summarize some anatomical facts and physiological results concerning the corticogeniculate loop, and then present a simplified model in which its function is to (usefully) mediate communication between cortical cells.

## 1.1   SOME ANATOMY AND PHYSIOLOGY

The LGN contains both principal cells, which project to cortex, and inhibitory interneurons. The projection to cortex sends collaterals into a sheet of inhibitory cells called the perigeniculate nucleus (PGN). PGN cells, in turn, project back to the LGN. The geniculocortical projection then proceeds into cortex, terminating principally in layers 4 and 6 in the cat [11, 12]. Areas 17, 18, and to a lesser extent, 19 are all innervated. Layer 6 cells in area 17 of the cat have particularly long, non-end-stopped receptive fields [2]. It is from layer 6 that the corticogeniculate projection back originates.[1] It, too, passes through the PGN, sending collaterals into it, and then contacts both principal cells and interneurons in the LGN, mostly in the more distal parts of their dendrites [10, 13]. Both the forward and the backward projection are retinotopically ordered.

Thus there is the possibility of both excitatory and inhibitory effects in the cortico-geniculate projection, which is principally what shall be used in the model.

The first attempts to study the physiology of the corticogeniculate projection involved inactivating cortex in some way (often cooling cortex) while observing geniculate responses to simple visual stimuli. The results were somewhat inconclusive: some investigators reported that the projection was excitatory, some that it was inhibitory, and still others that it had no observable effect at all. [1, 5, 9] Later studies have emphasized the need for using stimuli which optimally excite the cortical cells which project to the LGN; inactivating cortex should then make a significant difference in the inputs to geniculate cells. This has helped to reveal some effects: for example, LGN cells *with* corticogeniculate feedback are end-stopped (that is, respond much less to long bars than to short bars), while the end-stopping is quite clearly reduced when the cortical input is removed [8].

One study [13] has used cross-correlation analysis between cortical and geniculate cells to suggest that there is spatial structure in the corticogeniculate projection: an excitatory corticogeniculate interaction was found if cells had receptive field centers that were close to each other, while an inhibitory interaction was found if the centers were farther apart. However, the precise spatial structure of the projection remains unknown.

## 2   A FEEDBACK MODEL

I will now describe a simplified model of the LGN and the corticogeniculate loop. The very simple connection scheme shown in fig 1 originated in a suggestion by Christof Koch [3] that the long receptive fields in layer 6 might be used to facilitate contour completion at the LGN level. In the model, then, striate cortex simple cells feed back positively to the LGN, enhancing the conditions which gave rise to their firing. This reinforces, or completes, the oriented bar or edge patterns to which they are tuned. Assuming that the visual features of interest are for the most part oriented, while much of the noise in images may be isotropic and unoriented, enhancing the oriented features improves the signal-to-noise ratio.

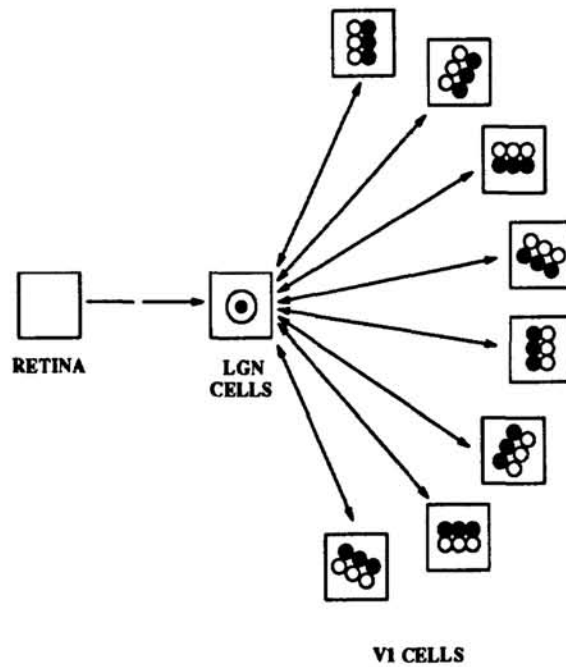

Figure 1: **Basic model connectivity**: A schematic diagram showing the connections between different pools of units in the single spatial frequency channel model. LGN cells first filter the image linearly through a center-surround filter ($\nabla^2 G$), the result of which is then passed through a sigmoid nonlinearity (tanh). (In the simulations presented here $G$ was a Gaussian with standard deviation 1.4 pixels.) V1 cells then provide oriented filtering, which is also passed through a nonlinearity (logistic; but see details in text) and fed back positively to the LGN to reinforce detected oriented edges. V1 cells excite LGN cells which have excitatory connections to them, and inhibit those that have inhibitory connections to them. Inhibition is implicitly assumed to be mediated by interneurons. (Note that there are no intracortical or intrageniculate connections: communication takes place entirely through the feedback loop.) See text for further details.

For simplicity, only striate cortex simple "edge-detecting" cells were modeled. Two models are presented. In the first one, all cortical cells have the same spatial frequency characteristics. In the second one, two channels, a high frequency channel and a low frequency channel, interact simultaneously.

## 2.1  SINGLE SPATIAL FREQUENCY CHANNEL MODEL

A schematic diagram of the model is shown in figure 1. The retina is used simply as an input layer. To each input position (pixel) in the retina there corresponds one LGN unit. Linear weights from the retina to the LGN implement a $\nabla^2 G$ filter, where $G(x,y)$ is a two-dimensional Gaussian. The LGN units then project to eight different pools of "orientation-tuned" cells in V1. Each of these pools has as many units as there are pixels in the input "retina". The weights in the projection forward to V1 represent eight rotations of the template shown in figure 2a, covering 360 degrees. This simulates basic orientation tuning in V1. Cortical cells then feed

back positively to the geniculus, using rotations of the template shown in fig 2(b).

The precise dynamics of the model are as follows: $R_i$ are real-valued retinal inputs, $L_i$ are geniculate unit outputs, and $V_i$ are cortical cell outputs. $G_{ji}$ represent weights from retina $\rightarrow$ LGN, $F_{ji}$ forward weights from LGN $\rightarrow$ V1, and $B_{ji}$ backward weights from V1 $\rightarrow$ LGN. $\alpha, \beta, \gamma, T_{C1}$ and $T_{C2}$ are all constants. For geniculate units:

$$\frac{dl_j}{dt} = -\gamma l_j + \sum_i G_{ji} R_i + \sum_k B_{jk} V_k \qquad L_j = \tanh(l_j)$$

While for cortical cell units:

$$\frac{dv_j}{dt} = -\alpha v_j + \sum_i F_{ji} L_i - \beta (\sum_i |F_{ji}| L_i)^2 \qquad V_j = \begin{cases} g(v_j - T_{C1}) & \text{if } v_j > T_{C2} \\ 0 & \text{otherwise} \end{cases}$$

Here $g()$ is the logistic function.

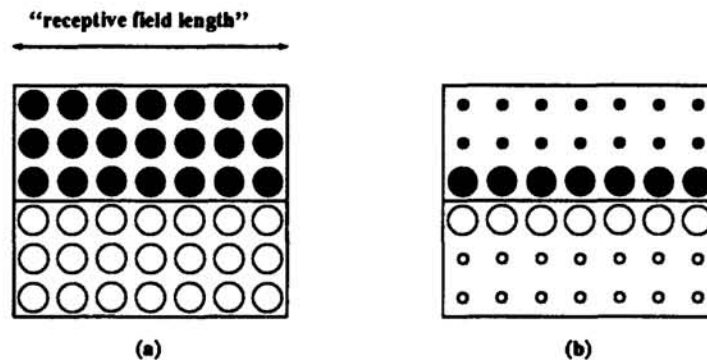

Figure 2: **Weights between the LGN and V1**. Figure 2(a): Forward weights, from the LGN to V1. Each circle represents the weight from a cell in the LGN; dark circles represent positive weights, light circles negative weights (assumed mediated by interneurons). The radius of each circle represents the strength of the corresponding weight. These weights create "edge-detecting" neurons in V1. Figure 2(b): Backwards weights, from V1 back to the LGN. Only cells close to the contrast edge receive strong feedback.

In the scheme described above many cortical cells have overlapping receptive fields, both in the forward projection from the geniculus and in the backwards projection from cortex. A cell which is reinforcing an edge within its receptive field will also partially reinforce the edge for retinotopically nearby cortical cells. For nearby cells with similar orientation tuning, the reinforcement will enhance their own firing; they will then enhance the firing of other, similar, cells farther along; and so on. That is, the overlapping feedback fields allow the edge detection process to *follow contours* (note that the process is tempered at the geniculate level by actual input from the retina). This is illustrated in figure 3.

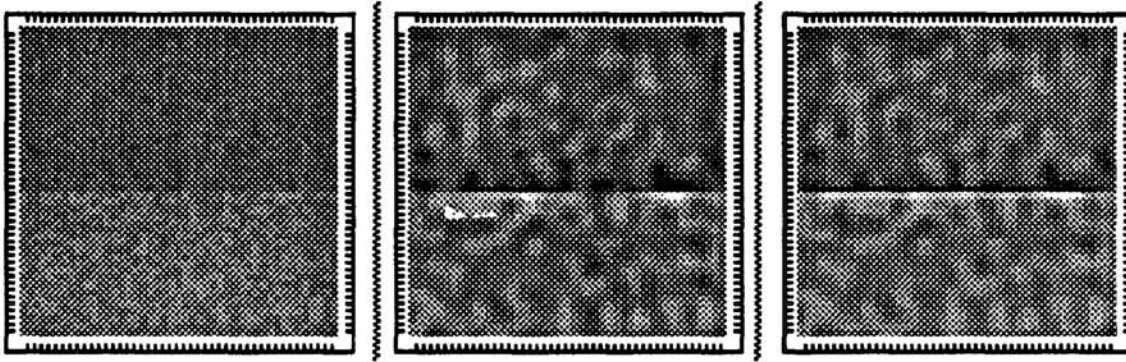

Figure 3: **Following contours:** This figure shows the effect on the LGN of the feedback enhancement. The image on the left is the retinal input: a very weak, noisy horizontal edge. The center image is the LGN after two iterations of the simulation. Note that initially only certain sectors of the edge are detected (and hence enhanced). The rightmost image is the LGN after 8 iterations: the enhanced region has spread to cover the entire edge through the effect of horizontally oriented, overlapping receptive fields. This is the final stable point of the dynamics.

## 2.2   MULTIPLE SPATIAL FREQUENCY CHANNELS MODEL

In the model described above the LGN is integrating and summarizing the information provided by each of the orientation-tuned pools of cortical cells.[2] It does so in a way which would easily extend to cover other types of cortical cells (bar or grating "detectors", or varying spatial frequency channels). To experiment simply with this possibility, an extra set of eight pools of orientation-tuned "edge-detecting" cortical cells was added. The new set's weights were similar to the original weights described above, except they had a "receptive field length" (see figure 2) of 3 pixels: the original set had a "receptive field length" of 9 pixels.

Thus one set was tuned for detecting short edges, while the other was tuned for detecting long edges. The effect of using both of these sets is illustrated in figure 4. Both sets interact nonlinearly to produce edge detection rather more robust than either set used alone: the image produced using both simultaneously is not a linear addition of those produced using each set separately. Note how little noise is accepted as an edge. The same model, running with the same parameters but more pixels, was also tested on a real image. This is shown in figure 5.

## 3   DISCUSSION ON CONNECTIVITY

A major function fulfilled by the LGN in this model is that of providing a communications pathway between cortical cells, both between cells of similar orientation but different location or spatial frequency tuning, and between cells of different orienta-

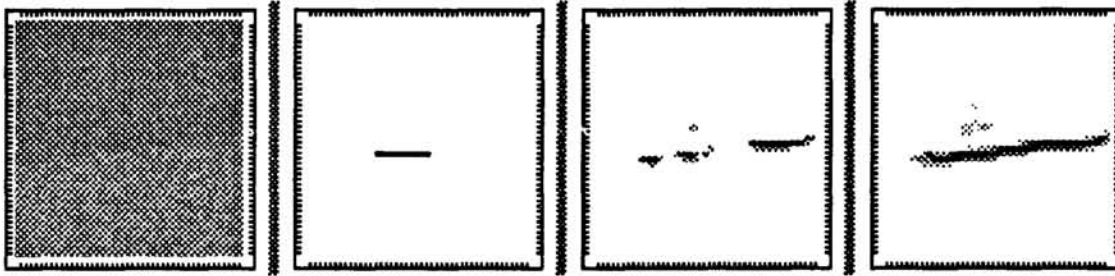

Figure 4: **Combined spatial frequency channels**: The leftmost image is the retinal input, a weak noisy edge. (The other three images are "summary outputs", obtained as follows: the model produces activations in many pools of cortical cell units; the activations from all V1 units corresponding to a particular retinotopic position are added together to form a real-valued number corresponding to that position; and this is then displayed as a grey-scale pixel. Since only "edge-detecting" units were used, this provides a rough estimate of the certainty of there being an edge at that point.) Second from left we see the summary output of the model after 20 iterations (by which time it has stabilized), using only the low spatial frequency channel. Only a single segment of the edge is detected. Third from left is the output after 20 iterations using only the high frequency channel. Only isolated, short, segments of the edge are detected. The rightmost image is the output using both channels simultaneously. Now the segments detected by the high frequency channel can combine with the original image to provide edges long enough for the low frequency channel to detect and complete into a single, long continuous edge.

tion tuning: for example, these last compete to reinforce their particular orientation preference on the geniculus. The model qualitatively shows that such a pathway, while mediated by a low-level representation like that of the LGN, can nevertheless be used effectively, producing contour-following and robust edge-detection. We must now ask whether such a function could not be performed without feedback. Clearly, it **could** be done without feedback to the LGN, purely through intracortical connections, since *any* feedback network can in principle be "unfolded in time" into a feedforward network which performs the same computation– provided we have enough units and connections available.

In other words, any suggested functional role for corticogeniculate feedback must not only include an account of the proposed computation performed, but also an account of why it is preferable to perform that computation through a feedback process, in terms of some efficiency measure (like the number of cells or synapses necessary, for example). There can be no other rationale, apart from fortuitous coincidence, for constructing an elaborate feedback mechanism to perform a computation that could just as well be done without it.

With this view in mind, it is worth re-stating that in this model any two cortical cells whose receptive fields overlap are connected (disynaptically) through the LGN. How many connections would we require in order to achieve similar communication if we only used direct connections between cortical orientation-tuned cells instead? In monkey, each cell's receptive field overlaps with approximately $10^6$ others [4]– thus,

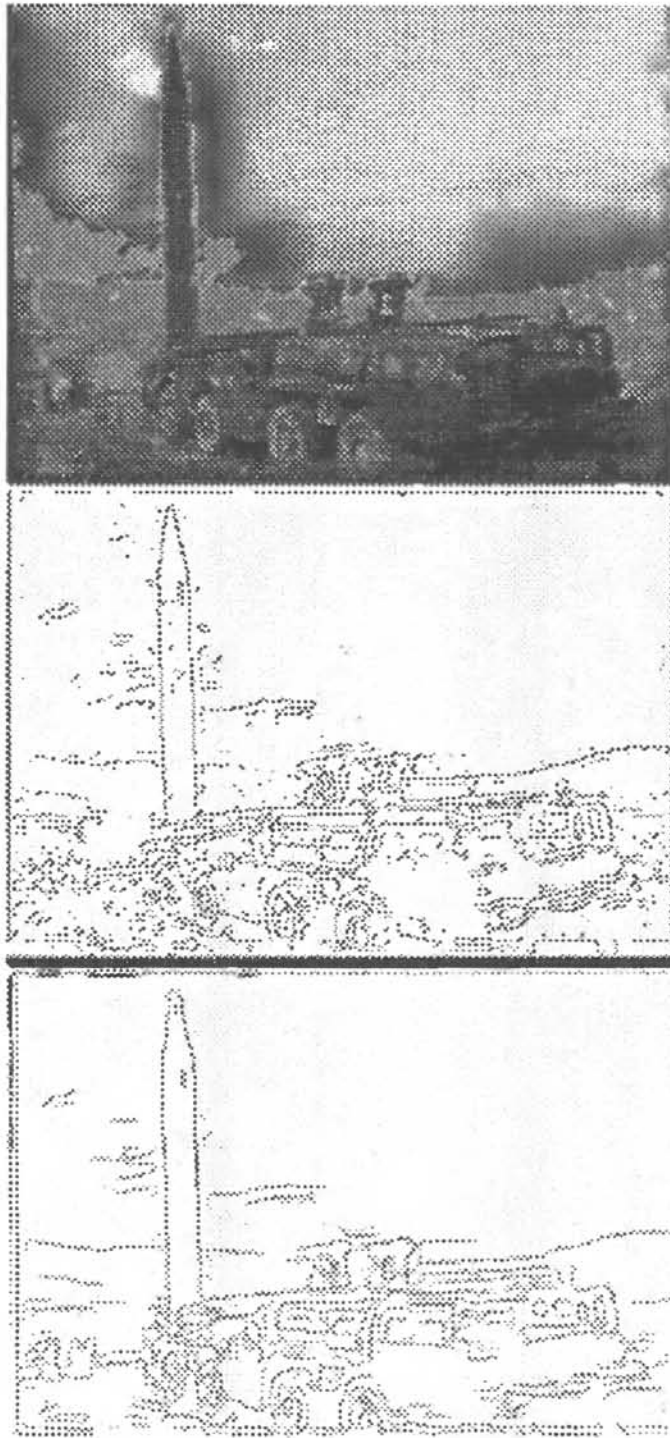

Figure 5: **A real image:** The top image is the retinal input. Stippling is due to printing only. The center image is that obtained through detecting the zero-crossings of $\nabla^2 G$. To reduce spurious edges, a minimum slope threshold was placed on the point of the zero-crossing below which edges were not accepted. The image shown here was the best that could be obtained through varying both the width of the Gaussian G and the slope threshold value. The last image shows the summary output from the model, using two simultaneous spatial frequency channels. Note how noise is reduced compared to the center image, straight lines are smoother, and resolution is not impaired, but is better in places (group of people at lower left, or "smoke stacks" atop launcher).

any cortical cell would need to synapse onto at least $10^6$ cells. If the information can be sent via the LGN, geniculate cell fan-out can reduce the number of necessary synapses by a significant factor. It is estimated that geniculate cells (in the cat) synapse onto at least 200 cortical cells (probably more) [6], reducing the number of necessary connections considerably.

## 4    BIOLOGY AND CONCLUSIONS

In section 1.1 I noted one important study [8] which established that corticogeniculate input reduces firing of geniculate cells for long bars; this is in direct contradiction to the prediction which would be made by this model, where the feedback enhances firing for long features (here, edges). Thus, the model does not agree with known physiology.

However, the model's value lies simply in clearly illustrating the possibility that feedback in a hierarchical processing scheme like the corticogeniculate loop can be utilized for robust, low-level pattern analysis, through the use of the cortex→LGN→cortex communications pathway. The possibility that a great deal of different types of information could be flowing through this pathway for this purpose should not be left unconsidered.

**Acknowledgements**

The author is supported by fellowships from the Parsons Foundation and from CONACYT (Mexico). Thanks are due to Michael Lyons for careful reading of the manuscript.

**References**

[1] Baker, F. H. and Malpeli, J. G. 1977 *Exp. Brain Res.* **29** pp. 433-444

[2] Gilbert, C.D. 1977, *J. Physiol.*, **268**, pp. 391-421

[3] Koch, C. 1992, personal communication.

[4] Hubel, D.H. and Wiesel, T. N. 1977, *Proc. R. Soc. Lond. (B)* **198** pp. 1-59

[5] Kalil, R. E. and Chase, R. 1970, *J. Neurophysiol.* **33** pp. 459-474

[6] Martin, K.A.C. 1988, *Q. J. Exp. Phy.* **73** pp. 637-702

[7] Mumford, D. 1991 *Biol. Cybern.* **65** pp. 135-145

[8] Murphy, P.C. and Sillito, A.M. 1987, *Nature* **329** pp. 727-729

[9] Richard, D. et. al. 1975, *Exp. Brain Res.* **22** pp. 235-242

[10] Robson, J. A. 1983, *J. Comp. Neurol.* **216** pp. 89-103

[11] Sherman, S. M. 1985, *Prog. in Psychobiol. and Phys. Psych.* **11** pp. 233-314

[12] Sherman, S.M. and Koch, C. 1986, *Exp. Brain Res.* **63** pp. 1-20

[13] Tsumoto, T. et. al. 1978, *Exp. Brain Res.* **32** pp. 345-364

## Footnotes

[1]In all areas innervated by the LGN.

[2]A function not unlike that suggested by Mumford [7], except that here the "experts" are extremely low-level orientation-tuned channels.
